# A Neural Probabilistic Language Model

**Yoshua Bengio,** **Réjean Ducharme and Pascal Vincent**
Département d'Informatique et Recherche Opérationnelle
Centre de Recherche Mathématiques
Université de Montréal
Montréal, Québec, Canada, H3C 3J7
{*bengioy,ducharme,vincentp*}*@iro.umontreal.ca*

## Abstract

A goal of statistical language modeling is to learn the joint probability function of sequences of words. This is intrinsically difficult because of the curse of dimensionality: we propose to fight it with its own weapons. In the proposed approach one learns simultaneously (1) a distributed representation for each word (i.e. a similarity between words) along with (2) the probability function for word sequences, expressed with these representations. Generalization is obtained because a sequence of words that has never been seen before gets high probability if it is made of words that are similar to words forming an already seen sentence. We report on experiments using neural networks for the probability function, showing on two text corpora that the proposed approach very significantly improves on a state-of-the-art trigram model.

## 1 Introduction

A fundamental problem that makes language modeling and other learning problems difficult is the *curse of dimensionality*. It is particularly obvious in the case when one wants to model the joint distribution between many discrete random variables (such as words in a sentence, or discrete attributes in a data-mining task). For example, if one wants to model the joint distribution of 10 consecutive words in a natural language with a vocabulary $V$ of size 100,000, there are potentially $100\,000^{10} - 1 = 10^{50} - 1$ free parameters.

A statistical model of language can be represented by the conditional probability of the next word given all the previous ones in the sequence, since $P(w_1^T) = \prod_{t=1}^{T} P(w_t|w_1^{t-1})$, where $w_t$ is the $t$-th word, and writing subsequence $w_i^j = (w_i, w_{i+1}, \cdots, w_{j-1}, w_j)$.

When building statistical models of natural language, one reduces the difficulty by taking advantage of word order, and the fact that temporally closer words in the word sequence are statistically more dependent. Thus, *n-gram* models construct tables of conditional probabilities for the next word, for each one of a large number of *contexts*, i.e. combinations of the last $n-1$ words: $P(w_t|w_1^{t-1}) \approx P(w_t|w_{t-n+1}^{t-1})$. Only those combinations of successive words that actually occur in the training corpus (or that occur frequently enough) are considered. What happens when a new combination of $n$ words appears that was not seen in the training corpus? A simple answer is to look at the probability predicted using smaller context size, as done in back-off trigram models [7] or in smoothed (or interpolated) trigram models [6]. So, in such models, how is generalization basically obtained from sequences of

words seen in the training corpus to new sequences of words? simply by looking at a short enough context, i.e., the probability for a long sequence of words is obtained by "gluing" very short pieces of length 1, 2 or 3 words that have been seen frequently enough in the training data. Obviously there is much more information in the sequence that precedes the word to predict than just the identity of the previous couple of words. There are at least two obvious flaws in this approach (which however has turned out to be very difficult to beat): first it is not taking into account contexts farther than 1 or 2 words, second it is not taking account of the "similarity" between words. For example, having seen the sentence The cat is walking in the bedroom in the training corpus should help us generalize to make the sentence A dog was running in a room almost as likely, simply because "dog" and "cat" (resp. "the" and "a", "room" and "bedroom", etc...) have similar semantics and grammatical roles.

## 1.1 Fighting the Curse of Dimensionality with its Own Weapons

In a nutshell, the idea of the proposed approach can be summarized as follows:

1. associate with each word in the vocabulary a distributed "feature vector" (a real-valued vector in $\mathbb{R}^m$), thereby creating a notion of similarity between words,
2. express the joint probability *function* of word sequences in terms of the feature vectors of these words in the sequence, and
3. learn simultaneously the word feature vectors and the parameters of that *function*.

The feature vector represents different aspects of a word: each word is associated with a point in a vector space. The number of features (e.g. $m = 30, 60$ or $100$ in the experiments) is much smaller than the size of the vocabulary. The probability function is expressed as a product of conditional probabilities of the next word given the previous ones, (e.g. using a multi-layer neural network in the experiment). This function has parameters that can be iteratively tuned in order to maximize the log-likelihood of the training data or a regularized criterion, e.g. by adding a weight decay penalty. The feature vectors associated with each word are learned, but they can be initialized using prior knowledge.

Why does it work? In the previous example, if we knew that dog and cat played similar roles (semantically and syntactically), and similarly for (the,a), (bedroom,room), (is,was), (running,walking), we could naturally generalize from The cat is walking in the bedroom to A dog was running in a room and likewise to many other combinations. In the proposed model, it will so generalize because "similar" words should have a similar feature vector, and because the probability function is a *smooth* function of these feature values, so a small change in the features (to obtain similar words) induces a small change in the probability: *seeing only one of the above sentences will increase the probability not only of that sentence but also of its combinatorial number of "neighbors" in sentence space (as represented by sequences of feature vectors).*

## 1.2 Relation to Previous Work

The idea of using neural networks to model high-dimensional discrete distributions has already been found useful in [3] where the joint probability of $Z_1 \cdots Z_n$ is decomposed as a product of conditional probabilities: $P(Z_1 = z_1, \cdots, Z_n = z_n) = \prod_i P(Z_i = z_i | g_i(z_{i-1}, z_{i-2}, \cdots, z_1))$, where $g(.)$ is a function represented by part of a neural network, and it yields parameters for expressing the distribution of $Z_i$. Experiments on four UCI data sets show this approach to work comparatively very well [3, 2]. The idea of a distributed representation for symbols dates from the early days of connectionism [5]. More recently, Hinton's approach was improved and successfully demonstrated on learning several symbolic relations [9]. The idea of using neural networks for language modeling is not new either, e.g. [8]. In contrast, here we push this idea to a large scale, and concentrate on learning a statistical model of the distribution of word sequences, rather than learning the role of words in a sentence. The proposed approach is also related to previous proposals

of character-based text compression using neural networks [11]. Learning a clustering of words [10, 1] is also a way to discover similarities between words. In the model proposed here, instead of characterizing the similarity with a discrete random or deterministic variable (which corresponds to a soft or hard partition of the set of words), we use a continuous real-vector for each word, i.e. a distributed feature vector, to indirectly represent similarity between words. The idea of using a vector-space representation for words has been well exploited in the area of *information retrieval* (for example see [12]), where vectorial feature vectors for words are learned on the basis of their probability of co-occurring in the same documents (Latent Semantic Indexing [4]). An important difference is that here we look for a representation for words that is helpful in representing compactly the probability distribution of word sequences from natural language text. Experiments indicate that learning jointly the representation (word features) and the model makes a big difference in performance.

## 2 The Proposed Model: two Architectures

The training set is a sequence $w_1 \cdots w_T$ of words $w_t \in V$, where the vocabulary $V$ is a large but finite set. The objective is to learn a good model $f(w_t, \cdots, w_{t-n}) = \hat{P}(w_t | w_1^{t-1})$, in the sense that it gives high out-of-sample likelihood. In the experiments, we will report the geometric average of $1/\hat{P}(w_t | w_1^{t-1})$, also known as *perplexity*, which is also the exponential of the average negative log-likelihood. The only constraint on the model is that for any choice of $w_1^{t-1}$, $\sum_{i=1}^{|V|} f(i, w_{t-1}, w_{t-n}) = 1$. By the product of these conditional probabilities, one obtains a model of the joint probability of any sequence of words.

The basic form of the model is described here. Refinements to speed it up and extend it will be described in the following sections. We decompose the function $f(w_t, \cdots, w_{t-n}) = \hat{P}(w_t | w_1^{t-1})$ in two parts:

1. A mapping $C$ from any element of $V$ to a real vector $C(i) \in \mathbb{R}^m$. It represents the "distributed feature vector" associated with each word in the vocabulary. In practice, $C$ is represented by a $|V| \times m$ matrix (of free parameters).

2. The probability function over words, expressed with $C$. We have considered two alternative formulations:

   (a) The **direct architecture**: a function $g$ maps a sequence of feature vectors for words in context $(C(w_{t-n}), \cdots, C(w_{t-1}))$ to a probability distribution over words in $V$. It is a vector function whose $i$-th element estimates the probability $\hat{P}(w_t = i | w_1^{t-1})$ as in figure 1. $f(i, w_{t-1}, \cdots, w_{t-n}) = g(i, C(w_{t-1}), \cdots, C(w_{t-n}))$. We used the "softmax" in the output layer of a neural net: $\hat{P}(w_t = i | w_1^{t-1}) = e^{h_i} / \sum_j e^{h_j}$, where $h_i$ is the neural network output score for word $i$.

   (b) The **cycling architecture**: a function $h$ maps a sequence of feature vectors $(C(w_{t-n}), \cdots, C(w_{t-1}), C(i))$ (i.e. including the context words and a candidate next word $i$) to a scalar $h_i$, and again using a softmax, $\hat{P}(w_t = i | w_1^{t-1}) = e^{h_i} / \sum_j e^{h_j}$. $f(w_t, w_{t-1}, \cdots, w_{t-n}) = g(C(w_t), C(w_{t-1}), \cdots, C(w_{t-n}))$. We call this architecture "cycling" because one repeatedly runs $h$ (e.g. a neural net), each time putting in input the feature vector $C(i)$ for a candidate next word $i$.

The function $f$ is a composition of these two mappings ($C$ and $g$), with $C$ being *shared* across all the words in the context. To each of these two parts are associated some parameters. The parameters of the mapping $C$ are simply the feature vectors themselves (represented by a $|V| \times m$ matrix $C$ whose row $i$ is the feature vector $C(i)$ for word $i$). The function $g$ may be implemented by a feed-forward or recurrent neural network or another parameterized function, with parameters $\theta$.

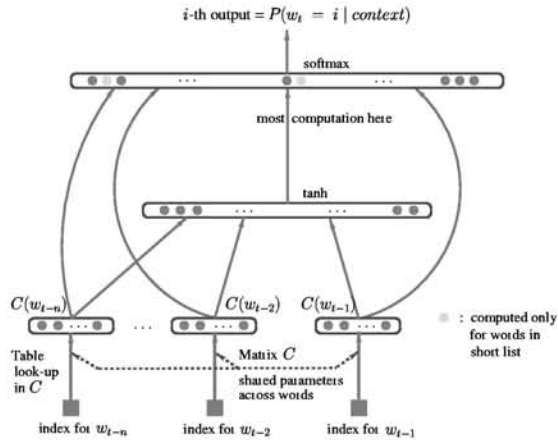

Figure 1: "Direct Architecture": $f(i, w_{t-1}, \cdots, w_{t-n}) = g(i, C(w_{t-1}), \cdots, C(w_{t-n}))$
where $g$ is the neural network and $C(i)$ is the $i$-th word feature vector.

Training is achieved by looking for $(\theta, C)$ that maximize the training corpus penalized log-likelihood: $L = \frac{1}{T} \sum_t \log p_{w_t}(C(w_{t-n}), \cdots, C(w_{t-1}); \theta) + R(\theta, C)$, where $R(\theta, C)$ is a regularization term (e.g. a weight decay $\lambda ||\theta||^2$, that penalizes slightly the norm of $\theta$).

## 3   Speeding-up and other Tricks

**Short list.** The main idea is to focus the effort of the neural network on a "short list" of words that have the highest probability. This can save much computation because in both of the proposed architectures the time to compute the probability of the observed next word scales almost linearly with the number of words in the vocabulary (because the scores $h_i$ associated with each word $i$ in the vocabulary must be computed for properly normalizing probabilities with the softmax). The idea of the speed-up trick is the following: instead of computing the actual probability of the next word, the neural network is used to compute the relative probability of the next word within that **short list**. The choice of the short list depends on the current context (the previous $n$ words). We have used our smoothed trigram model to pre-compute a short list containing the most probable next words associated to the previous two words. The conditional probabilities $\hat{P}(w_t = i|h_t)$ are thus computed as follows, denoting with $h_t$ the history (context) before $w_t$, and $L_t$ the short list of words for the prediction of $w_t$. If $i \in L_t$ then the probability is $\hat{P}_{NN}(w_t = i|w_t \in L_t, h_t)\hat{P}_{trigram}(w_t \in L_t|h_t)$, else it is $\hat{P}_{trigram}(w_t = i|h_t)$, where $\hat{P}_{NN}(w_t = i|w_t \in L_t, h_t)$ are the normalized scores of the words computed by the neural network, where the "softmax" is only normalized over the words in the short list $L_t$, and $\hat{P}_{trigram}(w_t \in L_t|h_t) = \sum_{i \in L_t} \hat{P}_{trigram}(i|h_t)$, with $\hat{P}_{trigram}(i|h_t)$ standing for the next-word probabilities computed by the smoothed trigram. Note that both $L_t$ and $\hat{P}_{trigram}(w_t \in L_t|h_t)$ can be pre-computed (and stored in a hash table indexed by the last two words).

**Table look-up for recognition.** To speed up application of the trained model, one can pre-compute in a hash table the output of the neural network, at least for the most frequent input contexts. In that case, the neural network will only be rarely called upon, and the average computation time will be very small. Note that in a speech recognition system, one needs only compute the *relative* probabilities of the acoustically ambiguous words in each context, also reducing drastically the amount of computations.

**Stochastic gradient descent.** Since we have millions of examples, it is important to converge within only a few passes through the data. For very large data sets, stochastic gradient descent convergence time seems to increase sub-linearly with the size of the data set (see experiments on Brown vs Hansard below). To speed up training using stochastic gradient

descent, we have found it useful to break the corpus in paragraphs and to randomly permute them. In this way, some of the non-stationarity in the word stream is eliminated, yielding faster convergence.

**Capacity control.** For the "smaller corpora" like Brown (1.2 million examples), we have found early stopping and weight decay useful to avoid over-fitting. For the larger corpora, our networks still under-fit. For the larger corpora, we have found **double-precision** computation to be very important to obtain good results.

**Mixture of models.** We have found improved performance by combining the probability predictions of the neural network with those of the smoothed trigram, with weights that were conditional on the frequency of the context (same procedure used to combine trigram, bigram, and unigram in the smoothed trigram).

**Initialization of word feature vectors.** We have tried both random initialization (uniform between -.01 and .01) and a "smarter" method based on a Singular Value Decomposition (SVD) of a very large matrix of "context features". These context features are formed by counting the frequency of occurrence of each word in each one of the most frequent contexts (word sequences) in the corpus. The idea is that "similar" words should occur with similar frequency in the same contexts. We used about 9000 most frequent contexts, and compressed these to 30 features with the SVD.

**Out-of-vocabulary words.** For an out-of-vocabulary word $w_t$ we need to come up with a feature vector in order to predict the words that follow, or predict its probability (that is only possible with the cycling architecture). We used as feature vector the weighted average feature vector of all the words in the short list, with the weights being the relative probabilities of those words: $E[C(w_t)|h_t] = \sum_i C(i)P(w_t = i|h_t)$.

# 4 Experimental Results

Comparative experiments were performed on the Brown and Hansard corpora. The Brown corpus is a stream of 1,181,041 words (from a large variety of English texts and books). The first 800,000 words were used for training, the following 200,000 for validation (model selection, weight decay, early stopping) and the remaining 181,041 for testing. The number of different words is $47,578$ (including punctuation, distinguishing between upper and lower case, and including the syntactical marks used to separate texts and paragraphs). Rare words with frequency $\leq 3$ were merged into a single token, reducing the vocabulary size to $|V| = 16,383$.

The Hansard corpus (Canadian parliament proceedings, French version) is a stream of about 34 million words, of which 32 millions (set A) was used for training, 1.1 million (set B) was used for validation, and 1.2 million (set C) was used for out-of-sample tests. The original data has $106,936$ different words, and those with frequency $\leq 10$ were merged into a single token, yielding $|V| = 30,959$ different words.

The benchmark against which the neural network was compared is an interpolated or smoothed trigram model [6]. Let $q_t = l(freq(w_{t-1}, w_{t-2}))$ represent the discretized frequency of occurrence of the context $(w_{t-1}, w_{t-2})$ (we used $l(x) = \lceil -\log((1 + x)/T) \rceil$ where $x$ is the frequency of occurrence of the context and $T$ is the size of the training corpus). A conditional mixture of the trigram, bigram, unigram and zero-gram was learned on the validation set, with mixture weights conditional on discretized frequency.

Below are measures of test set perplexity (geometric average of $1/\hat{P}(w_t|w_1^{t-1})$) for different models $\hat{P}$. Apparent convergence of the stochastic gradient descent procedure was obtained after around 10 epochs for Hansard and after about 50 epochs for Brown, with a learning rate gradually decreased from approximately $10^{-3}$ to $10^{-5}$. Weight decay of $10^{-4}$ or $10^{-5}$ was used in all the experiments (based on a few experiments compared on the validation set).

The **main result** is that **the neural network performs much better than the smoothed**

**trigram**. On Brown the best neural network system, according to validation perplexity (among different architectures tried, see below) yielded a perplexity of 258, while the smoothed trigram yields a perplexity of 348, which is about **35% worse**. This is obtained using a network with the direct architecture mixed with the trigram (conditional mixture), with 30 word features initialized with the SVD method, 40 hidden units, and $n = 5$ words of context. On Hansard, the corresponding figures are 44.8 for the neural network and 54.1 for the smoothed trigram, which is **20.7% worse**. This is obtained with a network with the direct architecture, 100 randomly initialized words features, 120 hidden units, and $n = 8$ words of context.

**More context is useful.** Experiments with the cycling architecture on Brown, with 30 word features, and 30 hidden units, varying the number of context words: $n = 1$ (like the bigram) yields a test perplexity of 302, $n = 3$ yields 291, $n = 5$ yields 281, $n = 8$ yields 279 (N.B. the smoothed trigram yields 348).

**Hidden units help.** Experiments with the direct architecture on Brown (with direct input to output connections), with 30 word features, 5 words of context, varying the number of hidden units: 0 yields a test perplexity of 275, 10 yields 267, 20 yields 266, 40 yields 265, 80 yields 265.

**Learning the word features jointly is important.** Experiments with the direct architecture on Brown (40 hidden units, 5 words of context), in which the word features initialized with the SVD method are kept fixed during training yield a test perplexity of 345.8 whereas if the word features are trained jointly with the rest of the parameters, the perplexity is 265.

**Initialization not so useful.** Experiments on Brown with both architectures reveal that the SVD initialization of the word features does not bring much improvement with respect to random initialization: it speeds up initial convergence (saving about 2 epochs), and yields a perplexity improvement of less than 0.3%.

**Direct architecture works a bit better.** The direct architecture was found about 2% better than the cycling architecture.

**Conditional mixture helps but even without it the neural net is better.** On Brown, the best neural net without the mixture yields a test perplexity of 265, the smoothed trigram yields 348, and their conditional mixture yields 258 (i.e., better than both). On Hansard the improvement is less: a neural network yielding 46.7 perplexity, mixed with the trigram (54.1), yields a mixture with perplexity 45.1.

## 5   Conclusions and Proposed Extensions

The experiments on two corpora, a medium one (1.2 million words), and a large one (34 million words) have shown that the proposed approach yields much better perplexity than a state-of-the-art method, the smoothed trigram, with differences on the order of 20% to 35%.

We believe that the main reason for these improvements is that the proposed approach allows to take advantage of the learned distributed representation to fight the curse of dimensionality with its own weapons: each training sentence informs the model about a combinatorial number of other sentences. Note that if we had a separate feature vector for each "context" (short sequence of words), the model would have much more capacity (which could grow like that of n-grams) but it would not naturally generalize between the many different ways a word can be used. A more reasonable alternative would be to explore language units other than words (e.g. some short word sequences, or alternatively some sub-word morphemic units).

There is probably much more to be done to improve the model, at the level of architecture, computational efficiency, and taking advantage of prior knowledge. An important priority of future research should be to evaluate and improve the speeding-up tricks proposed here, and find ways to increase capacity without increasing training time too much (to deal with

corpora with hundreds of millions of words). A simple idea to take advantage of temporal structure and extend the size of the input window to include possibly a whole paragraph, without increasing too much the number of parameters, is to use a time-delay and possibly recurrent neural network. In such a multi-layered network the computation that has been performed for small groups of consecutive words does not need to be redone when the network input window is shifted. Similarly, one could use a recurrent network to capture potentially even longer term information about the subject of the text.

A very important area in which the proposed model could be improved is in the use of prior linguistic knowledge: semantic (e.g. Word Net), syntactic (e.g. a tagger), and morphological (radix and morphemes). Looking at the word features learned by the model should help understand it and improve it. Finally, future research should establish how useful the proposed approach will be in applications to speech recognition, language translation, and information retrieval.

## Acknowledgments

The authors would like to thank Léon Bottou and Yann Le Cun for useful discussions. This research was made possible by funding from the NSERC granting agency.

## Footnotes

*Y.B. was also with AT&T Research while doing this research.

# References

[1] D. Baker and A. McCallum. Distributional clustering of words for text classification. In *SIGIR'98*, 1998.

[2] S. Bengio and Y. Bengio. Taking on the curse of dimensionality in joint distributions using neural networks. *IEEE Transactions on Neural Networks, special issue on Data Mining and Knowledge Discovery*, 11(3):550–557, 2000.

[3] Yoshua Bengio and Samy Bengio. Modeling high-dimensional discrete data with multi-layer neural networks. In S. A. Solla, T. K. Leen, and K-R. Mller, editors, *Advances in Neural Information Processing Systems 12*, pages 400–406. MIT Press, 2000.

[4] S. Deerwester, S.T. Dumais, G.W. Furnas, T.K. Landauer, and R.Harshman. Indexing by latent semantic analysis. *Journal of the American Society for Information Science*, 41(6):391–407, 1990.

[5] G.E. Hinton. Learning distributed representations of concepts. In *Proceedings of the Eighth Annual Conference of the Cognitive Science Society*, pages 1–12, Amherst 1986, 1986. Lawrence Erlbaum, Hillsdale.

[6] F. Jelinek and R. L. Mercer. Interpolated estimation of Markov source parameters from sparse data. In E. S. Gelsema and L. N. Kanal, editors, *Pattern Recognition in Practice*. North-Holland, Amsterdam, 1980.

[7] Slava M. Katz. Estimation of probabilities from sparse data for the language model component of a speech recognizer. *IEEE Transactions on Acoustics, Speech, and Signal Processing*, ASSP-35(3):400–401, March 1987.

[8] R. Miikkulainen and M.G. Dyer. Natural language processing with modular neural networks and distributed lexicon. *Cognitive Science*, 15:343–399, 1991.

[9] A. Paccanaro and G.E. Hinton. Extracting distributed representations of concepts and relations from positive and negative propositions. In *Proceedings of the International Joint Conference on Neural Network, IJCNN'2000*, Como, Italy, 2000. IEEE, New York.

[10] F. Pereira, N. Tishby, and L. Lee. Distributional clustering of english words. In *30th Annual Meeting of the Association for Computational Linguistics*, pages 183–190, Columbus, Ohio, 1993.

[11] Jürgen Schmidhuber. Sequential neural text compression. *IEEE Transactions on Neural Networks*, 7(1):142–146, 1996.

[12] H. Schutze. Word space. In S. J. Hanson, J. D. Cowan, and C. L. Giles, editors, *Advances in Neural Information Processing Systems 5*, pages pp. 895–902, San Mateo CA, 1993. Morgan Kaufmann.
